# Modelling transcriptional regulation using Gaussian processes

**Neil D. Lawrence**
School of Computer Science
University of Manchester, U.K.
neill@cs.man.ac.uk

**Guido Sanguinetti**
Department of Computer Science
University of Sheffield, U.K.
guido@dcs.shef.ac.uk

**Magnus Rattray**
School of Computer Science
University of Manchester, U.K.
magnus@cs.man.ac.uk

## Abstract

Modelling the dynamics of transcriptional processes in the cell requires the knowledge of a number of key biological quantities. While some of them are relatively easy to measure, such as mRNA decay rates and mRNA abundance levels, it is still very hard to measure the active concentration levels of the transcription factor proteins that drive the process and the sensitivity of target genes to these concentrations. In this paper we show how these quantities for a given transcription factor can be inferred from gene expression levels of a set of known target genes. We treat the protein concentration as a latent function with a Gaussian process prior, and include the sensitivities, mRNA decay rates and baseline expression levels as hyperparameters. We apply this procedure to a human leukemia dataset, focusing on the tumour repressor p53 and obtaining results in good accordance with recent biological studies.

## Introduction

Recent advances in molecular biology have brought about a revolution in our understanding of cellular processes. Microarray technology now allows measurement of mRNA abundance on a genome-wide scale, and techniques such as chromatin immunoprecipitation (ChIP) have largely unveiled the wiring of the cellular transcriptional regulatory network, identifying which genes are bound by which transcription factors. However, a full quantitative description of the regulatory mechanism of transcription requires the knowledge of a number of other biological quantities: first of all the concentration levels of active transcription factor proteins, but also a number of gene-specific constants such as the baseline expression level for a gene, the rate of decay of its mRNA and the sensitivity with which target genes react to a given transcription factor protein concentration. While some of these quantities can be measured (*e.g.* mRNA decay rates), most of them are very hard to measure with current techniques, and have therefore to be inferred from the available data. This is often done following one of two complementary approaches. One can formulate a large scale simplified model of regulation (for example assuming a linear response to protein concentrations) and then combine network architecture data and gene expression data to infer transcription factors' protein concentrations on a genome-wide scale. This line of research was started in [3] and then extended further to include gene-specific effects in [10, 11]. Alternatively, one can formulate a realistic model of a small subnetwork where few transcription factors regulate a small number of established target genes, trying to include the finer points of the dynamics of transcriptional regulation. In this paper we follow the second approach, focussing on the simplest subnetwork consisting of one tran-

scription factor regulating its target genes, but using a detailed model of the interaction dynamics to infer the transcription factor concentrations and the gene specific constants. This problem was recently studied by Barenco *et al.* [1] and by Rogers *et al.* [9]. In these studies, parametric models were developed describing the rate of production of certain genes as a function of the concentration of transcription factor protein at some specified time points. Markov chain Monte Carlo (MCMC) methods were then used to carry out Bayesian inference of the protein concentrations, requiring substantial computational resources and limiting the inference to the discrete time-points where the data was collected.

We show here how a Gaussian process model provides a simple and computationally efficient method for Bayesian inference of continuous transcription factor concentration profiles and associated model parameters. Gaussian processes have been used effectively in a number of machine learning and statistical applications [8] (see also [2, 6] for the work that is most closely related to ours). Their use in this context is novel, as far as we know, and leads to several advantages. Firstly, it allows for the inference of continuous quantities (concentration profiles) without discretization, therefore accounting naturally for the temporal structure of the data. Secondly, it avoids the use of cumbersome interpolation techniques to estimate mRNA production rates from mRNA abundance data, and it allows us to deal naturally with the noise inherent in the measurements. Finally, it greatly outstrips MCMC techniques in terms of computational efficiency, which we expect to be crucial in future extensions to more complex (and realistic) regulatory networks.

The paper is organised as follows: in the first section we discuss linear response models. These are simplified models in which the mRNA production rate depends linearly on the transcription factor protein concentration. Although the linear assumption is not verified in practice, it has the advantage of giving rise to an exactly tractable inference problem. We then discuss how to extend the formalism to model cases where the dependence of mRNA production rate on transcription factor protein concentration is not linear, and propose a MAP-Laplace approach to carry out Bayesian inference. In the third section we test our model on the leukemia data set studied in [1]. Finally, we discuss further extensions of our work. MATLAB code to recreate the experiments is available on-line.

# 1 Linear Response Model

Let the data set under consideration consist of $T$ measurements of the mRNA abundance of $N$ genes. We consider a linear differential equation that relates a given gene $j$'s expression level $x_j(t)$ at time $t$ to the concentration of the regulating transcription factor protein $f(t)$,

$$\frac{dx_j}{dt} = B_j + S_j f(t) - D_j x_j(t).$$ (1)

Here, $B_j$ is the basal transcription rate of gene $j$, $S_j$ is the sensitivity of gene $j$ to the transcription factor and $D_j$ is the decay rate of the mRNA. Crucially, the dependence of the mRNA transcription rate on the protein concentration (response) is linear. Assuming a linear response is a crude simplification, but it can still lead to interesting results in certain modelling situations. Equation (1) was used by Barenco *et al.* [1] to model a simple network consisting of the tumour suppressor transcription factor p53 and five of its target genes. We will consider more general models in section 2.

The equation given in (1) can be solved to recover

$$x_j(t) = \frac{B_j}{D_j} + k_j \exp(-D_j t) + S_j \exp(-D_j t) \int_0^t f(u) \exp(D_j u) \, du$$ (2)

where $k_j$ arises from the initial conditions, and is zero if we assume an initial baseline expression level $x_j(0) = B_j/D_j$.

We will model the protein concentration $f$ as a latent function drawn from a Gaussian process prior distribution. It is important to notice that equation (2) involves only linear operations on the function $f(t)$. This implies immediately that the mRNA abundance levels will also be modelled as a Gaussian process, and the covariance function of the marginal distribution $p(x_1, \ldots, x_N)$ can be worked out explicitly from the covariance function of the latent function $f$.

Let us rewrite equation (2) as

$$x_j\left(t\right) = \frac{B_j}{D_j} + L_j\left[f\right]\left(t\right)$$

where we have set the initial conditions such that $k_j$ in equation (2) is equal to zero and

$$L_j\left[f\right]\left(t\right) = S_j \exp\left(-D_j t\right) \int_0^t f\left(u\right) \exp\left(D_j u\right) du \qquad (3)$$

is the linear operator relating the latent function $f$ to the mRNA abundance of gene $j$, $x_j\left(t\right)$. If the covariance function associated with $f\left(t\right)$ is given by $k_{ff}\left(t, t'\right)$ then elementary functional analysis yields that

$$\mathrm{cov}\left(L_j\left[f\right]\left(t\right), L_k\left[f\right]\left(t'\right)\right) = L_j \otimes L_k\left[k_{ff}\right]\left(t, t'\right).$$

Explicitly, this is given by the following formula

$$k_{x_j x_k}\left(t, t'\right) = S_j S_k \exp\left(-D_j t - D_k t'\right) \int_0^t \exp\left(D_j u\right) \int_0^{t'} \exp\left(D_k u'\right) k_{ff}\left(u, u'\right) du' du. \qquad (4)$$

If the process prior over $f\left(t\right)$ is taken to be a squared exponential kernel,

$$k_{ff}\left(t, t'\right) = \exp\left(-\frac{\left(t - t'\right)^2}{l^2}\right),$$

where $l$ controls the width of the basis functions[1], the integrals in equation (4) can be computed analytically. The resulting covariances are obtained as

$$k_{x_j x_k}\left(t, t'\right) = S_j S_k \frac{\sqrt{\pi} l}{2}\left[h_{kj}\left(t', t\right) + h_{jk}\left(t, t'\right)\right] \qquad (5)$$

where

$$h_{kj}\left(t', t\right) = \frac{\exp\left(\gamma_k\right)^2}{D_j + D_k}\left\{\exp\left[-D_k\left(t' - t\right)\right]\left[\mathrm{erf}\left(\frac{t' - t}{l} - \gamma_k\right) + \mathrm{erf}\left(\frac{t}{l} + \gamma_k\right)\right]\right.$$
$$\left. -\exp\left[-\left(D_k t' + D_j\right)\right]\left[\mathrm{erf}\left(\frac{t'}{l} - \gamma_k\right) + \mathrm{erf}\left(\gamma_k\right)\right]\right\}.$$

Here $\mathrm{erf}(x) = \frac{2}{\sqrt{\pi}}\int_0^x \exp\left(-y^2\right) dy$ and $\gamma_k = \frac{D_k l}{2}$. We can therefore compute a likelihood which relates instantiations from all the observed genes, $\{x_j\left(t\right)\}_{j=1}^N$, through dependencies on the parameters $\{B_j, S_j, D_j\}_{j=1}^N$. The effect of $f\left(t\right)$ has been marginalised.

To infer the protein concentration levels, one also needs the "cross-covariance" terms between $x_j\left(t\right)$ and $f\left(t'\right)$, which is obtained as

$$k_{x_j f}\left(t, t'\right) = S_j \exp\left(-D_j t\right) \int_0^t \exp\left(D_j u\right) k_{ff}\left(u, t'\right) du. \qquad (6)$$

Again, this can be obtained explicitly for squared exponential priors on the latent function $f$ as

$$k_{x_j f}\left(t', t\right) = \frac{\sqrt{\pi} l S_j}{2}\exp\left(\gamma_j\right)^2 \exp\left[-D_j\left(t' - t\right)\right]\left[\mathrm{erf}\left(\frac{t' - t}{l} - \gamma_j\right) + \mathrm{erf}\left(\frac{t}{l} + \gamma_j\right)\right].$$

Standard Gaussian process regression techniques [see *e.g.* 8] then yield the mean and covariance function of the posterior process on $f$ as

$$\langle f\rangle_{\mathrm{post}} = K_{f\mathbf{x}} K_{\mathbf{xx}}^{-1}\mathbf{x}$$
$$K_{ff}^{\mathrm{post}} = K_{ff} - K_{f\mathbf{x}} K_{\mathbf{xx}}^{-1} K_{\mathbf{x}f} \qquad (7)$$

where $\mathbf{x}$ denotes collectively the $x_j\left(t\right)$ observed variables and capital $K$ denotes the matrix obtained by evaluating the covariance function of the processes on every pair of observed time points. The

model parameters $B_j$, $D_j$ and $S_j$ can be estimated by type II maximum likelihood. Alternatively, they can be assigned vague gamma prior distributions and estimated *a posteriori* using MCMC sampling.

In practice, we will allow the mRNA abundance of each gene at each time point to be corrupted by some noise, so that we can model the observations at times $t_i$ for $i = 1, \ldots, T$ as,

$$y_j(t_i) = x_j(t_i) + \epsilon_j(t_i) \tag{8}$$

with $\epsilon_j(t_i) \sim \mathcal{N}(0, \sigma_{ji}^2)$. Estimates of the confidence levels associated with each mRNA measurement can be obtained for Affymetrix microarrays using probe-level processing techniques such as the mmgMOS model of [4]. The covariance of the noisy process is simply obtained as $K_{\mathbf{yy}} = \Sigma + K_{\mathbf{xx}}$, with $\Sigma = \text{diag}\left(\sigma_{11}^2, \ldots, \sigma_{1T}^2, \ldots, \sigma_{N1}^2, \ldots, \sigma_{NT}^2\right)$.

## 2   Non-linear Response Model

While the linear response model presents the advantage of being exactly tractable in the important squared exponential case, a realistic model of transcription should account for effects such as saturation and ultrasensitivity which cannot be captured by a linear function. Also, all the quantities in equation (1) are positive, but one cannot constrain samples from a Gaussian process to be positive. Modelling the response of the transcription rate to protein concentration using a positive nonlinear function is an elegant way to enforce this constraint.

### 2.1   Formalism

Let the response of the mRNA transcription rate to transcription factor protein concentration levels be modelled by a nonlinear function $g$ with a target-specific vector $\boldsymbol{\theta}_j$ of parameters, so that,

$$\begin{aligned}
\frac{dx_j}{dt} &= B_j + g(f(t), \boldsymbol{\theta}_j) - D_j x_j \\
x_j(t) &= \frac{B_j}{D_j} + \exp(-D_j t) \int_0^t du\, g(f(u), \boldsymbol{\theta}_j) \exp(D_j u) \ ,
\end{aligned} \tag{9}$$

where we again set $x_j(0) = B_j/D_j$ and assign a Gaussian process prior distribution to $f(t)$. In this case the induced distribution of $x_j(t)$ is no longer a Gaussian process. However, we can derive the functional gradient of the likelihood and prior, and use this to learn the Maximum a Posteriori (MAP) solution for $f(t)$ and the parameters by (functional) gradient descent. Given noise-corrupted data $y_j(t_i)$ as above, the log-likelihood of the data $Y = \{y_j(t_i)\}$ is given by

$$p(Y|f, \{B_j, \theta_j, D_j, \Xi\}) = -\frac{1}{2} \sum_{i=1}^{T} \sum_{j=1}^{N} \left[ \frac{(x_j(t_i) - y_j(t_i))^2}{\sigma_{ji}^2} - \log\left(\sigma_{ji}^2\right) \right] - \frac{NT}{2} \log(2\pi) \tag{10}$$

where $\Xi$ denotes collectively the parameters of the prior covariance on $f$ (in the squared exponential case, $\Xi = l^2$). The functional derivative of the log-likelihood with respect to $f$ is then obtained as

$$\frac{\delta \log p(Y|f)}{\delta f(t)} = -\sum_{i=1}^{T} \Theta(t_i - t) \sum_{j=1}^{N} \frac{(x_j(t_i) - y_j(t_i))}{\sigma_{ji}^2} g'(f(t)) e^{-D_j(t_i - t)} \tag{11}$$

where $\Theta(x)$ is the Heaviside step function and we have omitted the model parameters for brevity. The negative Hessian of the log-likelihood with respect to $f$ is given by

$$\begin{aligned}
w(t, t') = -\frac{\delta^2 \log p(Y|f)}{\delta f(t) \delta f(t')} &= \sum_{i=1}^{T} \Theta(t_i - t) \delta(t - t') \sum_{j=1}^{N} \frac{(x_j(t_i) - y_j(t_i))}{\sigma_{ji}^2} g''(f(t)) e^{-D_j(t_i - t)} \\
&+ \sum_{i=1}^{T} \Theta(t_i - t) \Theta(t_i - t') \sum_{j=1}^{N} \sigma_{ji}^{-2} g'(f(t)) g'(f(t')) e^{-D_j(2t_i - t - t')}
\end{aligned} \tag{12}$$

where $g'(f) = \partial g / \partial f$ and $g''(f) = \partial^2 g / \partial f^2$.

## 2.2 Implementation

We discretise in time $t$ and compute the gradient and Hessian on a grid using approximate Riemann quadrature. In the simplest case, we choose a uniform grid $[t_p]$   $p = 1, \ldots, M$ so that $\Delta = t_p - t_{p-1}$ is constant. We write $\boldsymbol{f} = [f_p]$ to be the vector realisation of the function $f$ at the grid points. The gradient of the log-likelihood is then given by,

$$\frac{\partial \log p(Y|\boldsymbol{f})}{\partial f_p} = -\Delta \sum_{i=1}^{T} \Theta(t_i - t_p) \sum_{j=1}^{N} \frac{(x_j(t_i) - y_j(t_i))}{\sigma_{ji}^2} g'(f_p) \, e^{-D_j(t_i - t_p)} \qquad (13)$$

and the negative Hessian of the log-likelihood is,

$$W_{pq} = -\frac{\partial^2 \log p(Y|\boldsymbol{f})}{\partial f_p \partial f_q} = \delta_{pq} \Delta \sum_{i=1}^{T} \Theta(t_i - t_q) \sum_{j=1}^{N} \frac{(x_j(t_i) - y_j(t_i))}{\sigma_{ji}^2} g''(f_q) \, e^{-D_j(t_i - t_q)}$$

$$+ \Delta^2 \sum_{i=1}^{T} \Theta(t_i - t_p) \Theta(t_i - t_q) \sum_{j=1}^{N} \sigma_{ji}^{-2} g'(f_q) g'(f_p) \, e^{-D_j(2t_i - t_p - t_q)} \qquad (14)$$

where $\delta_{pq}$ is the Kronecker delta. In these and the following formulae $t_i$ is understood to mean the index of the grid point corresponding to the $i$th data point, whereas $t_p$ and $t_q$ correspond to the grid points themselves.

We can then compute the gradient and Hessian of the (discretised) un-normalised log posterior $\Psi(\boldsymbol{f}) = \log p(Y|\boldsymbol{f}) + \log p(\boldsymbol{f})$ [see 8, chapter 3]

$$\nabla \Psi(\boldsymbol{f}) = \nabla \log p(Y|\boldsymbol{f}) - K^{-1}\boldsymbol{f}$$
$$\nabla \nabla \Psi(\boldsymbol{f}) = -(W + K^{-1}) \qquad (15)$$

where $K$ is the prior covariance matrix evaluated at the grid points. These can be used to find the MAP solution $\hat{\boldsymbol{f}}$ using Newton's method. The Laplace approximation to the log-marginal likelihood is then (ignoring terms that do not involve model parameters)

$$\log p(Y) \simeq \log p(Y|\hat{\boldsymbol{f}}) - \tfrac{1}{2}\hat{\boldsymbol{f}}^T K^{-1}\hat{\boldsymbol{f}} - \tfrac{1}{2}\log|I + KW|. \qquad (16)$$

We can also optimise the log-marginal with respect to the model and kernel parameters. The gradient of the log-marginal with respect to the kernel parameters is [8]

$$\frac{\partial \log p(Y|\Xi)}{\partial \Xi} = \tfrac{1}{2}\hat{\boldsymbol{f}}^T K^{-1}\frac{\partial K}{\partial \Xi}K^{-1}\hat{\boldsymbol{f}} - \tfrac{1}{2}\mathrm{tr}\left((I + KW)^{-1}W\frac{\partial K}{\partial \Xi}\right) + \sum_p \frac{\partial \log p(Y|\Xi)}{\partial \hat{f}_p}\frac{\partial \hat{f}_p}{\partial \Xi} \qquad (17)$$

where the final term is due to the implicit dependence of $\hat{\boldsymbol{f}}$ on $\Xi$.

## 2.3 Example: exponential response

As an example, we consider the case in which

$$g(f(t), \theta_j) = S_j \exp(f(t)) \qquad (18)$$

which provides a useful way of constraining the protein concentration to be positive. Substituting equation (18) in equations (13) and (14) one obtains

$$\frac{\partial \log p(Y|\boldsymbol{f})}{\partial f_p} = -\Delta \sum_{i=1}^{T} \Theta(t_i - t_p) \sum_{j=1}^{N} \frac{(x_j(t_i) - y_j(t_i))}{\sigma_{ji}^2} S_j e^{f_p - D_j(t_i - t_p)}$$

$$W_{pq} = -\delta_{pq}\frac{\partial \log p(Y|\boldsymbol{f})}{\partial f_p} + \Delta^2 \sum_{i=1}^{T} \Theta(t_i - t_p) \Theta(t_i - t_q) \sum_{j=1}^{N} \sigma_{ji}^{-2} S_j^2 e^{f_p + f_q - D_j(2t_i - t_p - t_q)} .$$

The terms required in equation (17) are,

$$\frac{\partial \log p(Y|\Xi)}{\partial \hat{f}_p} = -(AW)_{pp} - \frac{1}{2}\sum_q A_{qq}W_{qp} \qquad \frac{\partial \hat{\boldsymbol{f}}}{\partial \Xi} = AK^{-1}\frac{\partial K}{\partial \Xi}\nabla \log p(Y|\hat{\boldsymbol{f}}),$$

where $A = (W + K^{-1})^{-1}$.

# 3  Results

To test the efficacy of our method, we used a recently published biological data set which was studied using a linear response model by Barenco *et al.* [1]. This study focused on the tumour suppressor protein p53. mRNA abundance was measured at regular intervals in three independent human cell lines using Affymetrix U133A oligonucleotide microarrays. The authors then restricted their interest to five known target genes of p53: *DDB2*, *p21*, *SESN1/hPA26*, *BIK* and *TNFRSF10b*. They estimated the mRNA production rates by using quadratic interpolation between any three consecutive time points. They then discretised the model and used MCMC sampling (assuming a log-normal noise model) to obtain estimates of the model parameters $B_j$, $S_j$, $D_j$ and $f(t)$. To make the model identifiable, the value of the mRNA decay of one of the target genes, p21, was measured experimentally. Also, the scale of the sensitivities was fixed by choosing p21's sensitivity to be equal to one, and $f(0)$ was constrained to be zero. Their predictions were then validated by doing explicit protein concentration measurements and growing mutant cell lines where the p53 gene had been knocked out.

## 3.1  Linear response analysis

We first analysed the data using the simple linear response model used by Barenco *et al.* [1]. Raw data was processed using the mmgMOS model of [4], which also provides estimates of the credibility associated with each measurement. Data from the different cell lines were treated as independent instantiations of $f$ but sharing the model parameters $\{B_j, S_j, D_j, \Xi\}$. We used a squared exponential covariance function for the prior distribution on the latent function $f$. The inferred posterior mean function for $f$, together with 95% confidence intervals, is shown in Figure 1(a). The pointwise estimates inferred by Barenco *et al.* are shown as crosses in the plot. The posterior mean function matches well the prediction obtained by Barenco *et al.*[2] Notice that the right hand tail of the inferred mean function shows an oscillatory behaviour. We believe that this is an artifact caused by the squared exponential covariance; the steep rise between time zero and time two forces the length scale of the function to be small, hence giving rise to wavy functions [see page 123 in 8]. To avoid this, we repeated the experiment using the "MLP" covariance function for the prior distribution over $f$ [12]. Posterior estimation cannot be obtained analytically in this case so we resorted to the MAP-Laplace approximation described in section 2. The MLP covariance is obtained as the limiting case of an infinite number of sigmoidal neural networks and has the following covariance function

$$k\left(t, t'\right) = \arcsin\left(\frac{wtt' + b}{\sqrt{\left(wt^2 + b + 1\right)\left(wt'^2 + b + 1\right)}}\right) \tag{19}$$

where $w$ and $b$ are parameters known as the weight and the bias variance. The results using this covariance function are shown in Figure 1(b). The resulting profile does not show the unexpected oscillatory behaviour and has tighter credibility intervals.

Figure 2 shows the results of inference on the values of the hyperparameters $B_j$, $S_j$ and $D_j$. The columns on the left, shaded grey, show results from our model and the white columns are the estimates obtained in [1]. The hyperparameters were assigned a vague gamma prior distribution ($a = b = 0.1$, corresponding to a mean of 1 and a variance of 10). Samples from the posterior distribution were obtained using Hybrid Monte Carlo [see *e.g.* 7]. The results are in good accordance with the results obtained by Barenco *et al.* Differences in the estimates of the basal transcription rates are probably due to the different methods used for probe-level processing of the microarray data.

## 3.2  Non-linear response analysis

We then used the non-linear response model of section 2 in order to constrain the protein concentrations inferred to be positive. We achieved this by using an exponential response of the transcription rate to the logged protein concentration. The inferred MAP solutions for the latent function $f$ are plotted in Figure 3 for the squared exponential prior (a) and for the MLP prior (b).

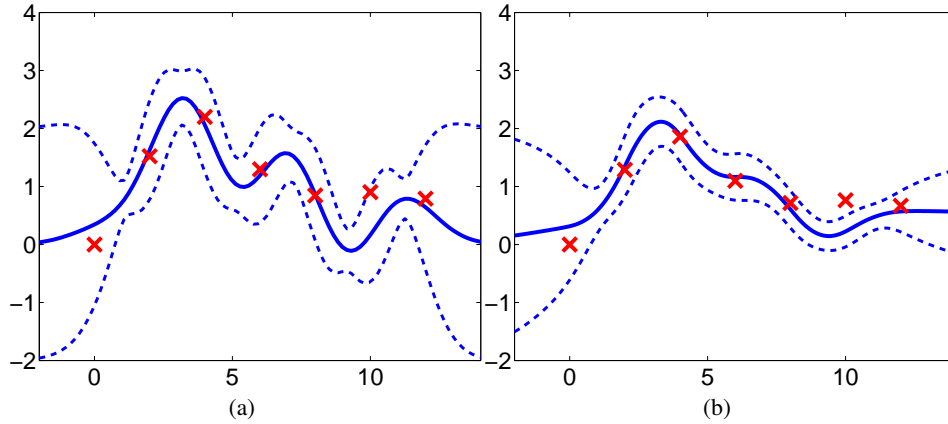

(a)                          (b)

Figure 1: Predicted protein concentration for p53 using a linear response model: (a) squared exponential prior on $f$; (b) MLP prior on $f$. Solid line is mean prediction, dashed lines are 95% credibility intervals. The prediction of Barenco *et al.* was pointwise and is shown as crosses.

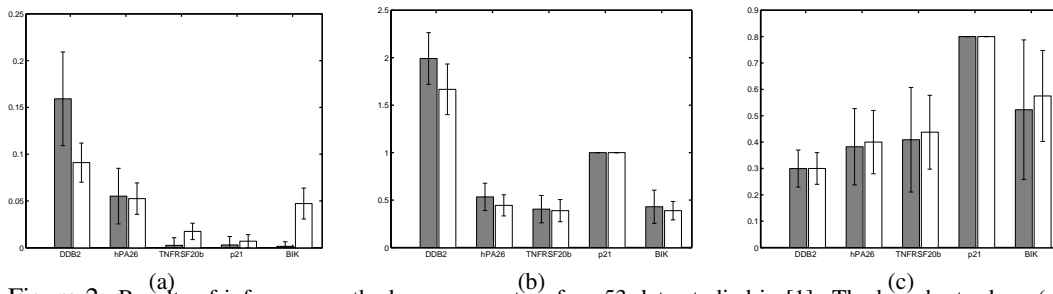

(a)                          (b)                          (c)

Figure 2: Results of inference on the hyperparameters for p53 data studied in [1]. The bar charts show (a) Basal transcription rates from our model and that of Barenco *et al.*. Grey are estimates obtained with our model, white are the estimates obtained by Barenco *et al.* (b) Similar for sensitivities. (c) Similar for decay rates.

# 4 Discussion

In this paper we showed how Gaussian processes can be used effectively in modelling the dynamics of a very simple regulatory network motif. This approach has many advantages over standard parametric approaches: first of all, there is no need to restrict the inference to the observed time points, and the temporal continuity of the inferred functions is accounted for naturally. Secondly, Gaussian processes allow noise information to be accounted for in a natural way. It is well known that biological data exhibits a large variability, partly because of technical noise (due to the difficulty to measure mRNA abundance for low expressed genes, for example), and partly because of the difference between different cell lines. Accounting for these sources of noise in a parametric model can be difficult (particularly when estimates of the derivatives of the measured quantities are required), while Gaussian Processes can incorporate this information naturally. Finally, MCMC parameter estimation in a discretised model can be computationally expensive due to the high correlations between variables. This is a consequence of treating the protein concentrations as parameters, and results in many MCMC iterations to obtain reliable samples. Parameter estimation can be achieved easily in our framework by type II maximum likelihood or by using efficient Monte Carlo sampling techniques only on the model hyperparameters.

While the results shown in the paper are encouraging, this is still a very simple modelling situation. For example, it is well known that transcriptional delays can play a significant role in determining the dynamics of many cellular processes [5]. These effects can be introduced naturally in a Gaussian process model; however, the data must be sampled at a reasonably high frequency in order for delays to become identifiable in a stochastic model, which is often not the case with microarray data sets. Another natural extension of our work would be to consider more biologically meaningful nonlinearities, such as the popular Michaelis-Menten model of transcription used in [9]. Finally, networks consisting of a single transcription factor are very useful to study small systems of particular interest such as p53. However, our ultimate goal would be to describe regulatory pathways consisting of

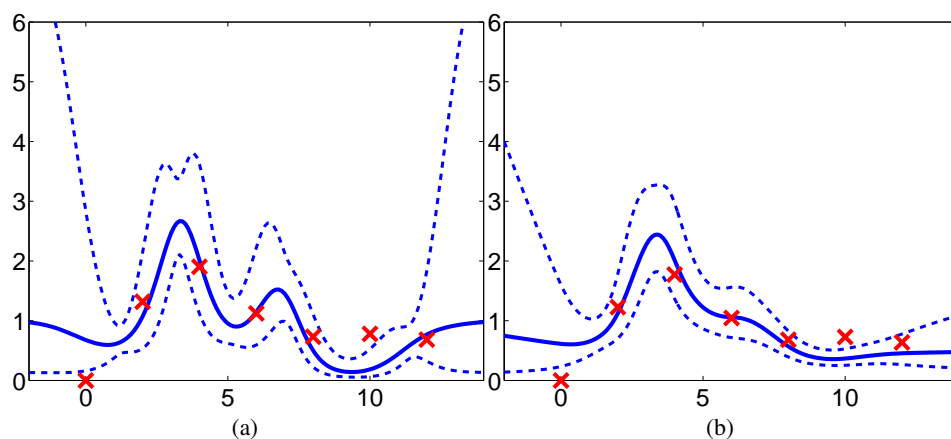

Figure 3: Predicted protein concentration for p53 using an exponential response: (a) shows results of using a squared exponential prior covariance on $f$; (b) shows results of using an MLP prior covariance on $f$. Solid line is mean prediction, dashed lines show 95% credibility intervals. The results shown are for $\exp(f)$, hence the asymmetry of the credibility intervals. The prediction of Barenco *et al.* was pointwise and is shown as crosses.

more genes. These can be dealt with in the general framework described in this paper, but careful thought will be needed to overcome the greater computational difficulties.

## Acknowledgements

We thank Martino Barenco for useful discussions and for providing the data. We gratefully acknowledge support from BBSRC Grant No BBS/B/0076X "Improved processing of microarray data with probabilistic models".

## Footnotes

[1]The scale of the process is ignored to avoid a parameterisation ambiguity with the sensitivities.

[2]Barenco *et al.* also constrained the latent function to be zero at time zero.

## References

[1] M. Barenco, D. Tomescu, D. Brewer, R. Callard, J. Stark, and M. Hubank. Ranked prediction of p53 targets using hidden variable dynamic modeling. *Genome Biology*, 7(3):R25, 2006.

[2] T. Graepel. Solving noisy linear operator equations by Gaussian processes: Application to ordinary and partial differential equations. In T. Fawcett and N. Mishra, editors, *Proceedings of the International Conference in Machine Learning*, volume 20, pages 234–241. AAAI Press, 2003.

[3] J. C. Liao, R. Boscolo, Y.-L. Yang, L. M. Tran, C. Sabatti, and V. P. Roychowdhury. Network component analysis: Reconstruction of regulatory signals in biological systems. *Proceedings of the National Academy of Sciences USA*, 100(26):15522–15527, 2003.

[4] X. Liu, M. Milo, N. D. Lawrence, and M. Rattray. A tractable probabilistic model for affymetrix probe-level analysis across multiple chips. *Bioinformatics*, 21(18):3637–3644, 2005.

[5] N. A. Monk. Unravelling nature's networks. *Biochemical Society Transactions*, 31:1457–1461, 2003.

[6] R. Murray-Smith and B. A. Pearlmutter. Transformations of Gaussian process priors. In J. Winkler, N. D. Lawrence, and M. Niranjan, editors, *Deterministic and Statistical Methods in Machine Learning*, volume 3635 of *Lecture Notes in Artificial Intelligence*, pages 110–123, Berlin, 2005. Springer-Verlag.

[7] R. M. Neal. *Bayesian Learning for Neural Networks*. Springer, 1996. Lecture Notes in Statistics 118.

[8] C. E. Rasmussen and C. K. Williams. *Gaussian Processes for Machine Learning*. MIT press, 2005.

[9] S. Rogers, R. Khanin, and M. Girolami. Model based identification of transcription factor activity from microarray data. In *Probabilistic Modeling and Machine Learning in Structural and Systems Biology*, Tuusula, Finland, 17-18th June 2006.

[10] C. Sabatti and G. M. James. Bayesian sparse hidden components analysis for transcription regulation networks. *Bioinformatics*, 22(6):739–746, 2006.

[11] G. Sanguinetti, M. Rattray, and N. D. Lawrence. A probabilistic dynamical model for quantitative inference of the regulatory mechanism of transcription. *Bioinformatics*, 22(14):1753–1759, 2006.

[12] C. K. I. Williams. Computation with infinite neural networks. *Neural Computation*, 10(5):1203–1216, 1998.